# Bayesian Model Comparison and Backprop Nets

**David J.C. MacKay***
Computation and Neural Systems
California Institute of Technology 139-74
Pasadena CA 91125
mackay@ras.phy.cam.ac.uk

## Abstract

The Bayesian model comparison framework is reviewed, and the Bayesian Occam's razor is explained. This framework can be applied to feedforward networks, making possible (1) objective comparisons between solutions using alternative network architectures; (2) objective choice of magnitude and type of weight decay terms; (3) quantified estimates of the error bars on network parameters and on network output. The framework also generates a measure of the effective number of parameters determined by the data.

The relationship of Bayesian model comparison to recent work on prediction of generalisation ability (Guyon *et al.*, 1992, Moody, 1992) is discussed.

## 1   BAYESIAN INFERENCE AND OCCAM'S RAZOR

In science, a central task is to develop and compare models to account for the data that are gathered. Typically, two levels of **inference** are involved in the task of data modelling. At the first level of inference, we assume that one of the models that we invented is true, and we fit that model to the data. Typically a model includes some free parameters; fitting the model to the data involves inferring what values those parameters should probably take, given the data. This is repeated for each model. The second level of inference is the task of model comparison. Here,

we wish to compare the models in the light of the data, and assign some sort of preference or ranking to the alternatives.[1]

For example, consider the task of interpolating a noisy data set. The data set could be interpolated using a splines model, polynomials, or feedforward neural networks. At the first level of inference, we find for each individual model the best fit interpolant (a process sometimes known as 'learning'). At the second level of inference we want to rank the alternative models and state for our particular data set that, for example, 'splines are probably the best interpolation model', or 'if the interpolant is modelled as a polynomial, it should probably be a cubic', or 'the best neural network for this data set has eight hidden units'.

Model comparison is a difficult task because it is not possible simply to choose the model that fits the data best: more complex models can always fit the data better, so the maximum likelihood model choice leads us inevitably to implausible over-parameterised models which generalise poorly. 'Occam's razor' is the principle that states that unnecessarily complex models should not be preferred to simpler ones. Bayesian methods automatically and quantitatively embody Occam's razor (Gull, 1988, Jeffreys, 1939), without the introduction of ad hoc penalty terms. Complex models are automatically self–penalising under Bayes' rule.

Let us write down Bayes' rule for the two levels of inference described above. Assume each model $\mathcal{H}_i$ has a vector of parameters $\mathbf{w}$. A model is defined by its functional form and two probability distributions: a 'prior' distribution $P(\mathbf{w}|\mathcal{H}_i)$ which states what values the model's parameters might plausibly take; and the predictions $P(D|\mathbf{w},\mathcal{H}_i)$ that the model makes about the data $D$ when its parameters have a particular value $\mathbf{w}$. Note that models with the same parameterisation but different priors over the parameters are therefore defined to be different models.

**1. Model fitting.** At the first level of inference, we assume that one model $\mathcal{H}_i$ is true, and we infer what the model's parameters $\mathbf{w}$ might be given the data $D$. Using Bayes' rule, the **posterior probability** of the parameters $\mathbf{w}$ is:

$$P(\mathbf{w}|D,\mathcal{H}_i) = \frac{P(D|\mathbf{w},\mathcal{H}_i)P(\mathbf{w}|\mathcal{H}_i)}{P(D|\mathcal{H}_i)} \tag{1}$$

In words:

$$\text{Posterior} = \frac{\text{Likelihood} \times \text{Prior}}{\text{Evidence}}$$

It is common to use gradient–based methods to find the maximum of the posterior, which defines the most probable value for the parameters, $\mathbf{w}_{\mathrm{MP}}$; it is then common to summarise the posterior distribution by the value of $\mathbf{w}_{\mathrm{MP}}$, and error bars on these best fit parameters. The error bars are obtained from the curvature of the posterior; writing the Hessian $\mathbf{A} = -\nabla\nabla \log P(\mathbf{w}|D,\mathcal{H}_i)$ and Taylor–expanding the log posterior with $\Delta\mathbf{w} = \mathbf{w} - \mathbf{w}_{\mathrm{MP}}$,

$$P(\mathbf{w}|D,\mathcal{H}_i) \simeq P(\mathbf{w}_{\mathrm{MP}}|D,\mathcal{H}_i) \exp\left(-\tfrac{1}{2}\Delta\mathbf{w}^{\mathrm{T}}\mathbf{A}\Delta\mathbf{w}\right) \tag{2}$$

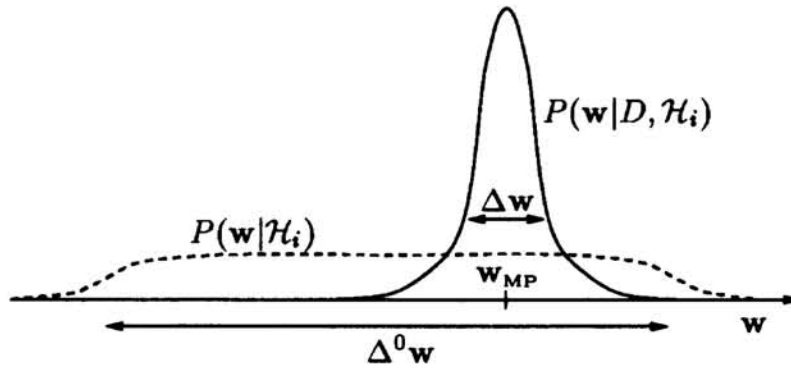

Figure 1: **The Occam factor**
This figure shows the quantities that determine the Occam factor for a hypothesis $\mathcal{H}_i$ having a single parameter $\mathbf{w}$. The prior distribution (dotted line) for the parameter has width $\Delta^0\mathbf{w}$. The posterior distribution (solid line) has a single peak at $\mathbf{w}_{\mathrm{MP}}$ with characteristic width $\Delta\mathbf{w}$. The Occam factor is $\frac{\Delta\mathbf{w}}{\Delta^0\mathbf{w}}$.

we see that the posterior can be locally approximated as a gaussian with covariance matrix (error bars) $\mathbf{A}^{-1}$.

**2. Model comparison.** At the second level of inference, we wish to infer which model is most plausible given the data. The posterior probability of each model is:

$$P(\mathcal{H}_i|D) \propto P(D|\mathcal{H}_i)P(\mathcal{H}_i) \qquad (3)$$

Notice that the objective data–dependent term $P(D|\mathcal{H}_i)$ is the evidence for $\mathcal{H}_i$, which appeared as the normalising constant in (1). The second term, $P(\mathcal{H}_i)$, is a 'subjective' prior over our hypothesis space. Assuming that we have no reason to assign strongly differing priors $P(\mathcal{H}_i)$ to the alternative models, **models $\mathcal{H}_i$ are ranked by evaluating the evidence.**

This concept is very general: the evidence can be evaluated for parametric and 'non–parametric' models alike; whether our data modelling task is a regression problem, a classification problem, or a density estimation problem, the evidence is the Bayesian's transportable quantity for comparing alternative models. In all these cases the evidence naturally embodies Occam's razor, as we will now see. The evidence is the normalising constant for equation (1):

$$P(D|\mathcal{H}_i) = \int P(D|\mathbf{w},\mathcal{H}_i)P(\mathbf{w}|\mathcal{H}_i)\,d\mathbf{w} \qquad (4)$$

For many problems, including interpolation, it is common for the posterior $P(\mathbf{w}|D,\mathcal{H}_i) \propto P(D|\mathbf{w},\mathcal{H}_i)P(\mathbf{w}|\mathcal{H}_i)$ to have a strong peak at the most probable parameters $\mathbf{w}_{\mathrm{MP}}$ (figure 1). Then the evidence can be approximated by the height of the peak of the integrand $P(D|\mathbf{w},\mathcal{H}_i)P(\mathbf{w}|\mathcal{H}_i)$ times its width, $\Delta\mathbf{w}$:

$$P(D|\mathcal{H}_i) \;\simeq\; \underbrace{P(D|\mathbf{w}_{\mathrm{MP}},\mathcal{H}_i)}_{} \; \underbrace{P(\mathbf{w}_{\mathrm{MP}}|\mathcal{H}_i)\,\Delta\mathbf{w}}_{} \qquad (5)$$

$$\text{Evidence} \;\simeq\; \text{Best fit likelihood} \quad \text{Occam factor}$$

Thus the evidence is found by taking the best fit likelihood that the model can achieve and multiplying it by an 'Occam factor' (Gull, 1988), which is a term with magnitude less than one that penalises $\mathcal{H}_i$ for having the parameter $\mathbf{w}$.

## Interpretation of the Occam factor

The quantity $\Delta\mathbf{w}$ is the posterior uncertainty in $\mathbf{w}$. Imagine for simplicity that the prior $P(\mathbf{w}|\mathcal{H}_i)$ is uniform on some large interval $\Delta^0\mathbf{w}$ (figure 1), so that $P(\mathbf{w}_{\text{MP}}|\mathcal{H}_i) = \frac{1}{\Delta^0\mathbf{w}}$; then

$$\text{Occam factor} = \frac{\Delta\mathbf{w}}{\Delta^0\mathbf{w}},$$

*i.e.* **the ratio of the posterior accessible volume of $\mathcal{H}_i$'s parameter space to the prior accessible volume** (Gull, 1988, Jeffreys, 1939). The log of the Occam factor can be interpreted as the amount of information we gain about the model $\mathcal{H}_i$ when the data arrive.

Typically, a complex or flexible model with many parameters, each of which is free to vary over a large range $\Delta^0\mathbf{w}$, will be penalised with a larger Occam factor than a simpler model. The Occam factor also penalises models which have to be finely tuned to fit the data. Which model achieves the greatest evidence is determined by a trade–off between minimising this natural complexity measure and minimising the data misfit.

## Occam factor for several parameters

If $\mathbf{w}$ is $k$-dimensional, and if the posterior is well approximated by a gaussian, the Occam factor is given by the determinant of the gaussian's covariance matrix:

$$P(D|\mathcal{H}_i) \simeq \underbrace{P(D|\mathbf{w}_{\text{MP}}, H_i)}_{\text{Evidence} \simeq \text{Best fit likelihood}} \underbrace{P(\mathbf{w}_{\text{MP}}|\mathcal{H}_i)(2\pi)^{k/2}\det^{-\frac{1}{2}}\mathbf{A}}_{\text{Occam factor}} \quad (6)$$

where $\mathbf{A} = -\nabla\nabla\log P(\mathbf{w}|D, \mathcal{H}_i)$, the Hessian which we already evaluated when we calculated the error bars on $\mathbf{w}_{\text{MP}}$. As the amount of data collected, $N$, increases, this gaussian approximation is expected to become increasingly accurate on account of the central limit theorem.

Thus Bayesian model selection is a simple extension of maximum likelihood model selection: **the evidence is obtained by multiplying the best fit likelihood by the Occam factor.** To evaluate the Occam factor all we need is the Hessian $\mathbf{A}$, if the gaussian approximation is good. Thus the Bayesian method of model comparison by evaluating the evidence is computationally no more demanding than the task of finding for each model the best fit parameters and their error bars.

## 2    THE EVIDENCE FOR NEURAL NETWORKS

Neural network learning procedures include a host of control parameters such as the number of hidden units and weight decay rates. These parameters are difficult to set because there is an Occam's razor problem: if we just set the parameters so as to minimise the error on the training set, we would be led to over–complex and under–regularised models which over–fit the data. Figure 2a illustrates this problem by showing the test error versus the training error of a hundred networks of varying complexity all trained on the same interpolation problem.

Of course if we had unlimited resources, we could compare these networks by measuring the error on an unseen test set or by similar cross–validation techniques. However these techniques may require us to devote a large amount of data to the test set, and may be computationally demanding. If there are several parameters like weight decay rates, it is preferable if they can be optimised on line.

Using the Bayesian framework, it is possible for all our data to have a say in both the model fitting and the model comparison levels of inference. We can rank alternative neural network solutions by evaluating the 'evidence'. Weight decay rates can also be optimised by finding the 'most probable' weight decay rate. Alternative weight decay schemes can be compared using the evidence. The evidence also makes it possible to compare neural network solutions with other interpolation models, for example, splines or radial basis functions, and to choose control parameters such as spline order or RBF kernel. The framework can be applied to classification networks as well as the interpolation networks discussed here. For details of the theoretical framework (which is due to Gull and Skilling (1989)) and for more complete discussion and bibliography, MacKay (1991) should be consulted.

## 2.1   THE PROBABILISTIC INTERPRETATION

Fitting a backprop network to a data set $D = \{\mathbf{x}, \mathbf{t}\}$ often involves minimising an objective function $M(\mathbf{w}) = \beta E_D(\mathbf{w}; D) + \alpha E_W(\mathbf{w})$. The first term is the data error, for example $E_D = \sum \frac{1}{2}(\mathbf{y} - \mathbf{t})^2$, and the second term is a regulariser (weight decay term), for example $E_W = \sum \frac{1}{2} w_h^2$. (There may be several regularisers with independent constants $\{\alpha_c\}$. The Bayesian framework also covers that case.) A model $\mathcal{H}$ has three components $\{\mathcal{A}, \mathcal{N}, \mathcal{R}\}$: The architecture $\mathcal{A}$ specifies the functional dependence of the input–output mapping on the network's parameters $\mathbf{w}$. The noise model $\mathcal{N}$ specifies the functional form of the data error. Within the probabilistic interpretation (Tishby *et al.*, 1989), the data error is viewed as relating to a likelihood, $P(D|\mathbf{w}, \beta, \mathcal{A}, \mathcal{N}) = \exp(-\beta E_D)/Z_D$. For example, a quadratic $E_D$ corresponds to the assumption that the distribution of errors between the data and the true interpolant is Gaussian, with variance $\sigma_\nu^2 = 1/\beta$. Lastly, the regulariser $\mathcal{R}$, with associated regularisation constant $\alpha$, is interpreted as specifying a prior on the parameters $\mathbf{w}$, $P(\mathbf{w}|\alpha, \mathcal{A}, \mathcal{R}) = \exp(-\alpha E_W)$. For example, the use of a plain quadratic regulariser corresponds to a Gaussian prior distribution for the parameters.

Given this probabilistic interpretation, interpolation with neural networks can then be decomposed into three levels of inference:

| 1 | Fitting a regularised model | $P(\mathbf{w}|D, \alpha, \beta, \mathcal{H}_i) = \dfrac{P(D|\mathbf{w}, \beta, \mathcal{H}_i) P(\mathbf{w}|\alpha, \mathcal{H}_i)}{P(D|\alpha, \beta, \mathcal{H}_i)}$ |
|---|---|---|
| 2a | Setting regularisation constants and estimating noise level | $P(\alpha, \beta|D, \mathcal{H}_i) = \dfrac{P(D|\alpha, \beta, \mathcal{H}_i) P(\alpha, \beta|\mathcal{H}_i)}{P(D|\mathcal{H}_i)}$ |
| 2 | Model comparison | $P(\mathcal{H}_i|D) \propto P(D|\mathcal{H}_i) P(\mathcal{H}_i)$ |

Both levels 2a and 2 require Occam's razor. For both levels the key step is to evaluate the evidence $P(D|\alpha, \beta, \mathcal{H})$, which, within the quadratic approximation

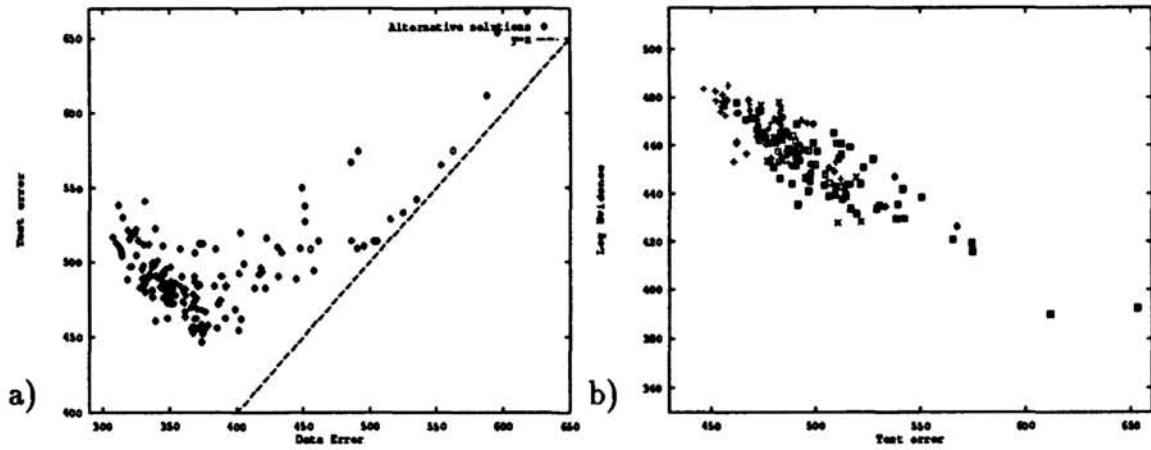

Figure 2: **The evidence solves the neural networks' Occam problem**
a) Test error vs. data error. Each point represents the performance of a single trained
neural network on the training set and on the test set. This graph illustrates the fact that
the best generalisation is not achieved by the models which fit the training data best.
b) Log Evidence vs. test error.

around $\mathbf{w}_{MP}$, is given by:

$$\log P(D|\alpha,\beta,\mathcal{H}) = -\alpha E_W^{MP} - \beta E_D^{MP} - \frac{1}{2}\log\det\mathbf{A} - \log Z_W(\alpha) - \log Z_D(\beta) + \frac{k}{2}\log 2\pi.$$

(7)

At level 2a we can find the most probable value for the regularisation constant $\alpha$
and noise level $1/\beta$ by differentiating (7) with respect to $\alpha$ and $\beta$. The result is

$$\chi_W^2 \equiv 2\alpha E_W = \gamma \qquad \text{and} \qquad \chi_D^2 \equiv 2\beta E_D = N - \gamma,$$

(8)

where $\gamma$ is 'the effective number of parameters determined by the data' (Gull, 1989),

$$\gamma = k - \alpha\text{Trace}\mathbf{A}^{-1} = \sum_{a=1}^{k}\frac{\lambda_a}{\lambda_a + \alpha},$$

(9)

where $\lambda_a$ are the eigenvalues of $\nabla\nabla\beta E_D$ in the natural basis of $E_W$. Each term
in the sum is a number between 0 and 1 which measures how well one parame-
ter is determined by the data rather than by the prior. The expressions (8), or
approximations to them, can be used to re–estimate weight decay rates on line.

The central quantity in the evidence and in $\gamma$ is the inverse hessian $\mathbf{A}^{-1}$, which
describes the error bars on the parameters $\mathbf{w}$. From this we can also obtain error
bars on the outputs of a network (Denker and Le Cun, 1991, MacKay, 1991). These
error bars are closely related to the predicted generalisation error calculated by
Levin *et al.*(1989). In (MacKay, 1991) the practical utility of these error bars is
demonstrated for both regression and classification networks.

Figure 2b shows the Bayesian 'evidence' for each of the solutions in figure 2a against
the test error. It can be seen that the correlation between the evidence and the
test error is extremely good. This good correlation depends on the model being
well–matched to the problem; when an inconsistent weight decay scheme was used
(forcing all weights to decay at the same rate), it was found that the correlation be-
tween the evidence and the test error was much poorer. Such comparisons between
Bayesian and traditional methods are powerful tools for human learning.

# 3   RELATION TO THEORIES OF GENERALISATION

The Bayesian 'evidence' framework assesses within a well–defined hypothesis space *how probable* a set of alternative models are. However, what we really want to know is how well each model is expected to generalise. Empirically, the correlation between the evidence and generalisation error is surprisingly good. But a theoretical connection linking the two is not yet established. Here, a brief discussion is given of similarities and differences between the evidence and quantities arising in recent work on prediction of generalisation error.

## 3.1   RELATION TO MOODY'S 'G.P.E.'

Moody's (1992) 'Generalised Prediction Error' is a generalisation of Akaike's 'F.P.E.' to non–linear regularised models. The F.P.E. is an estimator of generalisation error which can be derived without making assumptions about the distribution of errors between the data and true interpolant, and without assuming a known class to which the true interpolant belongs. The difference between F.P.E. and G.P.E. is that the total number of parameters $k$ in F.P.E. is replaced by an effective number of parameters, which is in fact identical to the quantity $\gamma$ arising in the Bayesian analysis (9). If $E_D$ is as defined earlier,

$$\text{G.P.E.} = \left( E_D + \sigma_\nu^2 \gamma \right) / N. \tag{10}$$

Like the log evidence, the G.P.E. has the form of the data error plus a term that penalises complexity. However, although the same quantity $\gamma$ arises in the Bayesian analysis, the Bayesian Occam factor does *not* have the same scaling behaviour as the G.P.E. term (see discussion below). And empirically, the G.P.E. is not always a good predictor of generalisation. The reason for this is that in the derivation of the G.P.E., it is assumed that the distribution over $\mathbf{x}$ values is well approximated by a sum of delta functions at the samples in the training set. This is equivalent to assuming test samples will be drawn only at the $\mathbf{x}$ locations at which we have already received data. This assumption breaks down for over–parameterised and over–flexible models. An additional distinction that between the G.P.E. and the evidence framework is that the G.P.E. is defined for regression problems only; the evidence can be evaluated for regression, classification and density estimation models.

## 3.2   RELATION TO THE EFFECTIVE V–C DIMENSION

Recent work on 'structural risk minimisation' (Guyon *et al.*, 1992) utilises empirical expressions of the form:

$$E_{\text{gen}} \simeq E_D/N + c_1 \frac{\log(N/\gamma) + c_2}{N/\gamma} \tag{11}$$

where $\gamma$ is the 'effective V–C dimension' of the model, and is identical to the quantity arising in (9). The constants $c_1$ and $c_2$ are determined by experiment. The structural risk theory is currently intended to be applied only to nested families of classification models (hence the absence of $\beta$: $E_D$ is dimensionless) with monotonic effective V–C dimension, whereas the evidence can be evaluated for any models. However, it is very interesting that the scaling behaviour of this expression (11) is

identical to the scaling behaviour of the log evidence (7), subject to the following assumptions. Assume that the value of the regularisation constant satisfies (8). Assume furthermore that the significant eigenvalues ($\lambda_a > \alpha$) scale as $\lambda_a \sim N\alpha/\gamma$ (It can be confirmed that this scaling is obtained for example in the interpolation models consisting of a sequence of steps of independent heights, as we vary the number of steps.) Then it can be shown that the scaling of the log evidence is:

$$-\log P(D|\alpha, \beta, \mathcal{H}) \sim \beta E_D^{\mathrm{MP}} + \frac{1}{2}\left(\gamma \log(N/\gamma) + \gamma\right) \tag{12}$$

(Readers familiar with MDL will recognise the dominant $\frac{\gamma}{2}\log N$ term; MDL and Bayes are identical.) Thus the scaling behaviour of the log evidence is identical to the structural risk minimisation expression (11), if $c_1 = \frac{1}{2}$ and $c_2 = 1$. I. Guyon (personal communication) has confirmed that the empirically determined values for $c_1$ and $c_2$ are indeed close to these Bayesian values. It will be interesting to try and understand and develop this relationship.

## Acknowledgements

This work was supported by studentships from Caltech and SERC, UK.

## Footnotes

*Current address: Darwin College, Cambridge CB3 9EU, U.K.

[1]Note that both levels of inference are distinct from *decision theory*. The goal of inference is, given a defined hypothesis space and a particular data set, to assign probabilities to hypotheses. Decision theory chooses between alternative actions on the basis of these probabilities so as to minimise the expectation of a 'loss function'.

## References

J.S. Denker and Y. Le Cun (1991). 'Transforming neural-net output levels to probability distributions', in *Advances in neural information processing systems 3*, ed. R.P. Lippmann *et al.*, 853–859, Morgan Kaufmann.

S.F. Gull (1988). 'Bayesian inductive inference and maximum entropy', in *Maximum Entropy and Bayesian Methods in science and engineering, vol. 1: Foundations*, G.J. Erickson and C.R. Smith, eds., Kluwer.

S.F. Gull (1989). 'Developments in Maximum entropy data analysis', in J. Skilling, ed., 53–71.

I. Guyon, V.N. Vapnik, B.E. Boser, L.Y. Bottou and S.A. Solla (1992). 'Structural risk minimization for character recognition', this volume.

H. Jeffreys (1939). *Theory of Probability*, Oxford Univ. Press.

E. Levin, N. Tishby and S. Solla (1989). 'A statistical approach to learning and generalization in layered neural networks', in *COLT '89: 2nd workshop on computational learning theory*, 245–260.

D.J.C. MacKay (1991) 'Bayesian Methods for Adaptive Models', Ph.D. Thesis, Caltech. Also 'Bayesian interpolation', 'A practical Bayesian framework for backprop networks', 'Information–based objective functions for active data selection', to appear in *Neural computation*. And 'The evidence framework applied to classification networks', submitted to *Neural computation*.

J.E. Moody (1992). 'Generalization, regularization and architecture selection in nonlinear learning systems', this volume.

N. Tishby, E. Levin and S.A. Solla (1989). 'Consistent inference of probabilities in layered networks: predictions and generalization', in *Proc. IJCNN*, Washington.